# TEMPORAL REPRESENTATIONS IN A CONNECTIONIST SPEECH SYSTEM

Erich J. Smythe
207 Greenmanville Ave, #6
Mystic, CT 06355

## ABSTRACT

SYREN is a connectionist model that uses temporal information in a speech signal for syllable recognition. It classifies the rates and directions of formant center transitions, and uses an adaptive method to associate transition events with each syllable. The system uses explicit spatial temporal representations through delay lines. SYREN uses implicit parametric temporal representations in formant transition classification through node activation onset, decay, and transition delays in sub–networks analogous to visual motion detector cells. SYREN recognizes 79% of six repetitions of 24 consonant-vowel syllables when tested on unseen data, and recognizes 100% of its training syllables.

## INTRODUCTION

Living organisms exist in a dynamic environment. Problem solving systems, both natural and synthetic, must relate and interpret events that occur over time. Although connectionist models are based on metaphors from the brain, few have been designed to capture temporal and sequential information common to even the most primitive nervous systems. Yet some of the most popular areas of application of these models, including speech recognition, vision, and motor control, require some form of temporal processing.

The variation in a speech signal contains considerable information. Changes in format location or other acoustic parameters (Delattre, *et al.*, 1955; Pols and Schouten, 1982) can determine the identity of constituents of speech, even when segmentation information is obscure. Speech recognition systems have shown good results when they incorporate some temporal information (Waible, *et al.*, 1988, Anderson, *et al.*, 1988). Successful speech systems must incorporate temporal processing.

Natural organisms have sensory organs that are continuously updated and can do only limited buffering of input stimuli. Synthetic implementations can buffer their input, transforming time into space. Often the size and complexity of the input representations place limits on the amount of input that can be buffered, especially when data is coming from hundreds or thousands of sensors, and other methods must be found to integrate temporal information.

This paper describes SYREN (SYllable REcognition Network), a connectionist network that incorporates various temporal representations for consonant–vowel (CV) syllable recognition by the classification of formant center transitions. Input is presented sequentially, one time slice at a time. The network is described, including the temporal processing used in formant transition classification, learning, and syllable recognition. The results of syllable recognition experiments are discussed in the final section.

## TEMPORAL REPRESENTATIONS

Various types of temporal representations may be used to incorporate time in connectionist models. They range from explicit spatial representations where time is converted into space, to implicit parametric representations where time is incorporated using network computational parameters. Spatiotemporal representations are a middle ground combining the two extremes. The categories represent a continuum rather than absolute distinctions. Several of these types are found in SYREN.

## EXPLICIT SPATIAL REPRESENTATIONS

In a purely spatial representation temporal information is preserved by spreading time steps over space through the network topology. These representations include input buffers, delay lines, and recurrent networks.

Fixed input buffers allow interaction between time slices of input. Parts of the network are copied to represent states at particular time slices. Other methods use sliding input buffers in the form of a queue. Tapped delay lines and delay filters are means of spreading network node activations over time. Composed of chains of network nodes or delay functions, they can preserve the sequential structure of information. A value on a connection from a delay line represents events that have occurred in the past. Delay lines and filters have been used in speech recognition systems by Waible, *et al.* (1988), and Tank and Hopfield (1987).

Recurrent networks are similar to delay lines in that information is preserved by propagating activation through the network. They can store information indefinitely or generate potentially infinite sequences of behaviors through feedback cycles, whereas delay lines without cycles are limited by their fixed length. Recurrent networks pose problems for learning, although researchers are working on recurrent back propagation networks (Jordan, 1988).

Spatial representations are good for explicitly preserving sequences of events, and can simplify the learning of temporal patterns. Resource constraints place a limit on the size of fixed length buffers and delay lines, however. Input data from thousands of sensors place limits on the length of time represented in the buffer, and may not be able to retain information long enough to be of use. Fixed input buffers may introduce edge effects. Interaction is lost between the edges of the buffer and data from preceding and succeeding buffers unless the input is properly segmented. Long delay lines may be computationally expensive as well.

## SPATIOTEMPORAL REPRESENTATIONS

Implicit parametric methods represent time in connectionist models by the behavior of network nodes. State information stored in individual nodes allows more complex activation functions and the accumulation of statistical information. This method may be used to regulate the flow of activation in the network, provide a trace of previous activation, and learn from data separated in time.

Adjusting the parameters of functions such as the interactive activation equation of McClelland and Rumelhart (1982) can control the strength of input, affecting the rate that activation reaches saturation. This leads to pulse trains used in synchronization. Variations in decay parameters control the duration of an activation trace.

State and statistical information is useful in learning. Eligibility traces from classical conditioning models provide decaying memory of past connection activation. Temporally weighted averages may be used for weight computations.

Spatiotemporal representations combine implicit parametric representations with explicit spatial representations. These include the regulation of propagation time and pulse trains through parameter adjustment. Gating behavior that controls the flow of activation through a network is another spatiotemporal method.

## SYREN DESCRIPTION

SYREN is a connectionist model that incorporates temporal processing in isolated syllable recognition using formant center transitions. Formant center tracts are presented in 5 ms time slices. Input nodes are updated once per time slice. The network classifies the rates and directions of formant transitions. Transition data are used by an adaptive network to associate transition patterns with syllables. A recognition network uses output of the adaptive network to identify a syllable. Complete details of the system may be found in Smythe (1988).

## DATA CORPUS

Input data consist of formant centers from five repetitions of twenty-four consonant-vowel syllables (the stop consonants /b, d, g/ paired with the vowels /ii, ey, ih, eh, ae, ah, ou, uu/), and an averaged set of each of the five repetitions from work performed by Kewley Port (1982). Each repetition is presented as a binary matrix with a row representing frequency in 20 Hz units, and a column representing time in 5 ms slices. The matrix is given to the input units one column at a time. A '1' in a cell of a matrix represents a formant center at a particular frequency during a particular time slice.

## FORMANT TRANSITION CLASSIFICATION

In the first stage of processing SYREN determines the rate and direction of formant center transitions. Formant transition detectors are subnetworks designed to respond to transitions of one of six rates in either rising or falling directions,

and also to steady–state events. The method used is motivated by a mechanism for visual motion detection in the retina that combines interactions between sub-units of a dendritic tree and shunting, veto inhibition (Koch *et al*, 1982). Formant motion is analogous to visual motion, and formant transitions are treated as a one dimensional case of visual motion.

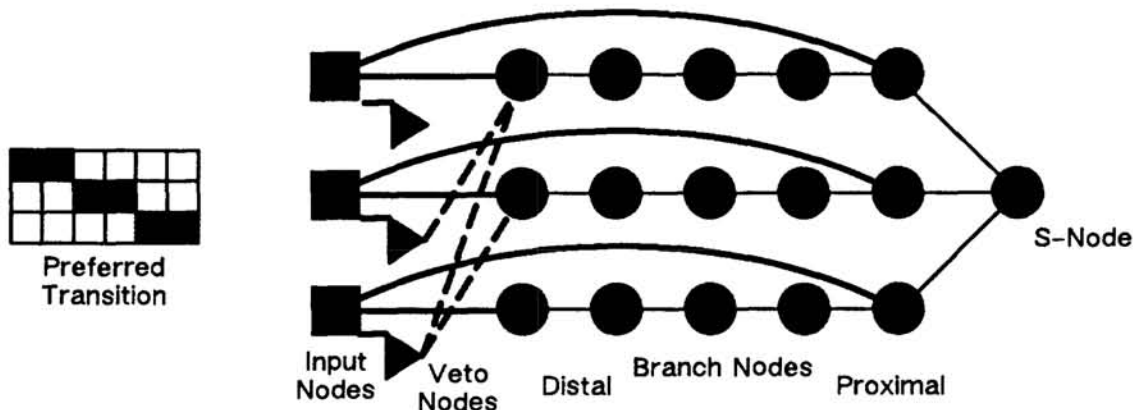

**Figure 1**. Formant transition detector subnetwork and its preferred descending transition type. The vertical axis is frequency (one row for each input unit) and the horizontal axis is time in 5 ms slices.

A detector subnetwork for a slow transition is shown in figure 1, along with its preferred transition. Branch nodes are analogous to dendritic subunits, and serve as activation transmission lines. Their activation is computed by the equation:

$$a_i^{t+1} = a_i^t(1-\theta) + net_i^t(1-a_i^t)$$

Where $a$ is the activation of unit $i$ at time $t$, $net$ is the weighted input, $t$ is an update cycle (there are 7 updates per time slice), and $\theta$ is a decay constant. Input to a branch node drives the activation to a maximum value, the rate of which is determined by the strength of the input. In the absence of input the activation decays to 0.

For the preferred direction, input nodes are activated for two time slices (10 ms) in order from top to bottom. An input node causes the activation of the most distal branch node to rise to a maximum value. This in turn causes the next node to activate, slightly delayed with respect to the first, and so on for the rest of the branch. This results in a pulse of activation flowing along the branch with a transmission delay of roughly one time slice (7 update cycles) from the distal to the proximal end. The most proximal branch node also has a connection to the input node. This connection serves to prime the node for slower transitions. Activation from an input node that lasts for only one time slice will decay in the proximal branch node before the activation from the distal region arrives. If input is present for two time steps the extra activation from the input connection primes the node, quickly driving it to a maximal value when the distal activation arrives.

An S-node provides the output of the detector. It computes a sigmoid squash function and fires (a sudden increase in activation) when sufficient activation is in the proximal branch nodes. For this particular detector, if the transition is too fast (*i.e.* one time step for each input unit) the proximal nodes will not attain a high enough activation; if the transition is too slow (*i.e.* three time steps for each input unit) activation on proximal branch nodes from earlier time steps will have decayed before the transition is complete. This architecture is tuned to a slower transition by increasing the transmission time on the branches by varying the connection weights, and by reducing the decay rate by lowering the decay constant. This illustrates the use of parametric manipulations to control temporal behavior in for rate sensitivity.

Veto inhibition is used in this detector for direction selectivity. Veto nodes provide inhibition and are activated by input nodes, and use the interactive activation equation for a decaying memory. Had the transition in figure 1. been in the opposite direction, activation from previous time slices on a veto connection would prevent the input node from activating its distal branch node, preventing the flow of activation and the firing of the S-node. Here a veto connection acts as a gate, serving to select input for processing.

Detectors are constructed for faster transitions by shortening the transmission lines and by using veto connections for rate sensitivity. A transition detector for a faster transition is shown in figure 2. Here the receptive field is larger, and veto connections are used to select transitions that skip one input unit at each time slice. Veto connections are still used for direction selectivity. Detectors for even faster transitions are created by widening the receptive field and increasing the number of veto connections for rate sensitivity.

Detectors are designed to respond to a specific transition type and not to respond to the transitions of other detectors. They will respond to transitions with rates between their own and the next type of detector. For slower transitions the firing of two detectors indicates an intermediate rate. For faster transitions special detectors are designed to fire for only one precise rate by eliminating some of the branches. Different firing patterns of precise and more general detectors distinguish rates. This gives a very fine rate sensitivity throughout the range of transitions.

Detector networks are copied to span the entire frequency range with overlapping receptive fields. This yields an array of S-nodes for each transition type, giving excellent spatial resolution of the frequency range. There are 200 S-nodes for each detector type, each signaling a transition that starts and ends at a particular frequency unit.

## ADAPTIVE NETWORK

The adaptive network learns to associate patterns of formant transitions with specific syllables. To do this it must be able to store at least part of the unfolding patterns or else it is forced to respond to information from only one time slice.

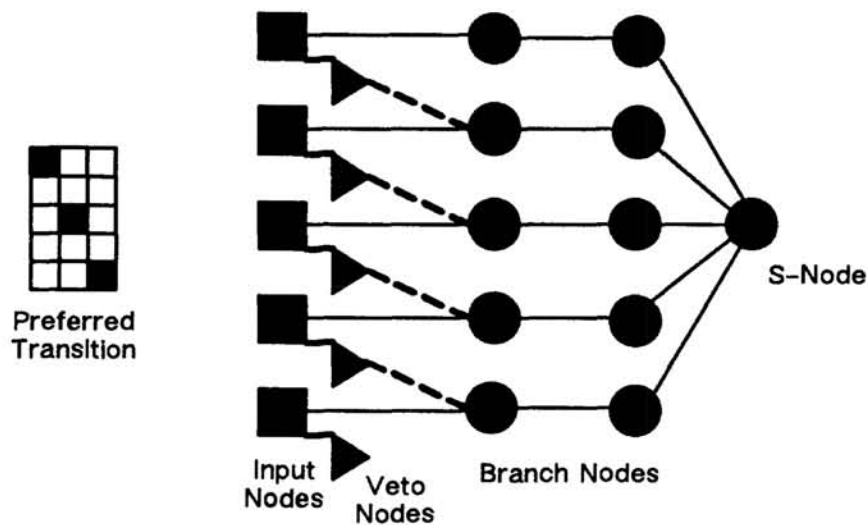

**Figure 2**. Formant transition detector subnetwork for a faster transition. Only the veto connections used for rate sensitivity are shown.

The learning algorithm must also deal with past activation histories of connections or else it can only learn from one time slice. The network accomplishes this through tapped delay lines and decaying eligibility traces.

There are twenty-four nodes in the adaptive network, each assigned to one syllable. It is a single layer network, trained using a hybrid supervised learning algorithm that merges Widrow-Hoff type learning with a classical conditioning model (Sutton and Barto, 1987).

## Storage of temporal patterns

Tapped delay lines are used to briefly store sequences of formant transition patterns. S-nodes from each transition detector are connected to a tapped delay line of five nodes. Each delay node simply passes on its S-node's activation value once per 5 ms time slice, allowing the delay matrix to store 25 ms (five time slices) of transition patterns.

The delay matrix consists of delay lines for each transition detector at each receptive field. Adaptive nodes are connected to every node in the delay matrix. The delay lines do not perform input buffering; information in the delay matrix has been subject to one level of processing. The amount of information stored (the length of the delay line) is limited by efficiency considerations.

## Adaptive Algorithm

Nodes in the adaptive network compute their activation using a sigmoid squash function and adjust their weights according to the equation:

$$w_{ij}^{t+1} = w_{ij}^t + a(z_i^t - s_i^t)e_j^t$$

where $w$ is the weight from a connection from node $j$ to node $i$ at time $t$, $\alpha$ is a learning constant, $z$ is the expected value of node $i$, $s$ is the weighted sum of the connections of node $i$, and $e$ is the exponentially decaying canonical eligibility of

connection *j*. The eligibility constant gives some variation in the exact timing of transition patterns, allowing limited time warping between training and testing.

## FINAL RECOGNITION NETWORK

The adaptive network is not perfect and results in a number of false alarm errors. Many of these are eliminated by using firing patterns of other adaptive nodes. For example, a node that consistently misfires on one syllable could be blocked by the firing of the correct node for that syllable. Adaptive nodes are connected to a veto recognition network. Since an adaptive node may fire at any time (and at different times) throughout input presentation, delay lines are used to preserve patterns of adaptive node behavior, and veto inhibition is used to block false alarms. Connections in the veto network are enabled or disabled after training. Clearly this is an *ad hoc* solution, but it suggests the use of representations that are distributed both spatially and temporally.

## RESULTS AND DISCUSSION

In each experiment syllable repetitions were divided into mutually exclusive training and testing sets. A training cycle consisted of one presentation of each member of the training set. In both experiments the networks were trained until adequate performance was achieved, usually after four to ten training cycles.

In the first experiment the network was trained on the five raw repetitions and tested on the averaged set. It achieved 92% recognition on the testing set and 100% recognition on the training set. The network had two miss errors on the training set.

In the second experiment, the network was trained on four of the raw repetitions and tested on the fifth. Five separate training runs were performed to test the network on each repetition. The network achieved 76% recognition on the testing set for all training runs, and 100% recognition on the training set.

In all experiments most of the adaptive nodes responded when there was transition information in the delay matrix. Many responded when both transition and steady–state information was present, using clues from both the consonant and the vowel. This situation occurs only briefly for each formant, since the delay matrix holds information for 5 time slices, and it takes four time slices to signal a steady–state event. Transition information will be at the end of the delay matrix while steady–state is at the beginning. Many nodes were strongly inhibited in the absence of transition information even for their correct syllable, although they had fired earlier in the data presentation.

## CONCLUSIONS

We have shown how different temporal representations and processing methods are used in a connectionist model for syllable recognition. Hybrid connectionist architectures with only slightly more elaborate processing methods can classify acoustic motion and associate sequences of transition events with syllables. The

system is not designed as a general speech recognition system, especially since the accurate measurement of formant center frequencies is impractical. Other signal processing techniques, such as spectral peak estimation, can be used without changes in the architecture. This could provide information to a larger speech recognition system.

SYREN was influenced by a neurophysiological model for visual motion detection, and shows how knowledge from one processing modality is applied to other problems. The merging of ideas from real nervous systems with existing techniques can add to the connectionist tool kit, resulting in more powerful processing systems.

## Acknowledgments

This research was performed at Indiana University Computer Science Department as part of the author's Ph.D. thesis. The author would like to thank committee members John Barnden and Robert Port for their help and direction, and Donald Lee and Peter Brodeur for their assistance in preparing the manuscript.

## References

Delattre, P. C., Liberman, A. M., Cooper, F. S., 1955, "Acoustic loci and transitional cues for stop consonants," *J. Acous. Soc. Am.*, **27**, 769–773.

Jordan, M. I., 1986 "Serial order: A parallel distributed processing approach," ICS Report 8604, UCSD, San Diego.

Kewley-Port, D., 1982, "Measurement of formant transitions in naturally produced consonant–vowel syllables," *J. Acous. Soc. Am*, **72**, 379–389.

Koch, C., Poggio, T., Torre, V., 1982, "Retinal ganglion cells: A functional interpretation of dendritic morphology," *Phil. Trans. R. Soc. Lon.: Series B*, **298**, 227–264.

McClelland, J. L., Rumelhart, D. E., 1982, "An interactive activation model of context effects in letter perception," *Psychological Review*, **88**, 375–407.

Pols, L. C. W., Schouten, M. F. H., 1982, "Perceptual relevance of coarticulation," in: Carlson, R., and Grandstrom, B., *The Representation of Speech in the Peripheral Auditory System*, Elsevier, 203–208.

Anderson, S., Merrill, J.W.L, Port, R., 1988, "Dynamic speech characterization with recurrent networks," Indiana University Dept. of Computer Science TR. No. 258, Bloomington, In.

Smythe, E. J., 1988, "Temporal computation in connectionist models," Indiana University Dept. of Computer Science TR. No. 251, Bloomington, In.

Sutton, R. S., Barto, A. G., 1987, "A temporal difference model of classical conditioning," GTE TR87-509.2.

Tank, D. W., Hopfield, J. J., 1987, "Concentrating information in time," *Proceedings of the IEEE Conference on Neural Networks*, San Diego, IV-455-468.

Waible, A., Hanazawa, T., Hinton, G., Shikana, K, Lang, K, 1988, "Phoneme recognition: Neural networks vs. Hidden Markov Models," *Proc. Int. Conf. Acoustics, Speech, and Signal Processing*, 107–110.